# Promoting Poor Features to Supervisors: Some Inputs Work Better as Outputs

**Rich Caruana**
JPRC and
Carnegie Mellon University
Pittsburgh, PA 15213
caruana@cs.cmu.edu

**Virginia R. de Sa**
Sloan Center for Theoretical Neurobiology and
W. M. Keck Center for Integrative Neuroscience
University of California, San Francisco CA 94143
desa@phy.ucsf.edu

## Abstract

In supervised learning there is usually a clear distinction between inputs and outputs — inputs are what you will measure, outputs are what you will predict from those measurements. This paper shows that the distinction between inputs and outputs is not this simple. Some features are more useful as extra *outputs* than as *inputs*. By using a feature as an output we get more than just the case values but can learn a mapping from the other inputs to that feature. For many features this mapping may be more useful than the feature value itself. We present two regression problems and one classification problem where performance improves if features that could have been used as inputs are used as extra outputs instead. This result is surprising since a feature used as an output is not used during testing.

## 1 Introduction

The goal in supervised learning is to learn functions that map inputs to outputs with high predictive accuracy. The standard practice in neural nets is to use all features that will be available for the test cases as inputs, and use as outputs only the features to be predicted.

Extra features available for training cases that *won't* be available during testing can be used as extra *outputs* that often benefit the original output[2][5]. Other ways of adding information to supervised learning through outputs include hints[1], tangent-prop[7], and EBNN[8]. In unsupervised learning it has been shown that inputs arising from different modalities can provide supervisory signals (outputs for the other modality) to each other and thus aid learning [3][6].

If outputs are so useful, and since any input could be used as an output, would some inputs be more useful as outputs? Yes. In this paper we show that in supervised backpropagation learning, some features are more useful as outputs than as inputs. This is surprising since using a feature as an output only extracts information from it during training; during testing it is not used.

This paper uses the following terms: The Main Task is the output to be learned. The goal is to improve performance on the Main Task. Regular Inputs are the features provided as inputs in all experiments. Extra Inputs (Extra Outputs) are the extra features when used as inputs (outputs). STD is standard backpropagation using the Regular Inputs as inputs and the Main Task as outputs. STD+IN uses the Extra Features as Extra Inputs to learn the Main Task. STD+OUT uses the Extra Features, but as Extra Outputs learned in parallel with the Main Task, using just the Regular Inputs as inputs.

## 2   Poorly Correlated Features

This section presents a simple synthetic problem where it is easy to see why using a feature as an extra output is better than using that same feature as an extra input.

Consider the following function:

$$F1(A,B) = SIGMOID(A+B), \qquad SIGMOID(x) = 1/(1 + e^{(-x)})$$

The STD net in Figure 1a has 20 inputs, 16 hidden units, and one output. We use backpropagation on this net to learn F1(). A and B are uniformly sampled from the interval [-5,5]. The network's input is binary codes for A and B. The range [-5,5] is discretized into $2^{10}$ bins and the binary code of the resulting bin number is used as the input coding. The first 10 input units receive the code for A and the second 10 that for B. The target output is the unary real (unencoded) value F1(A,B).

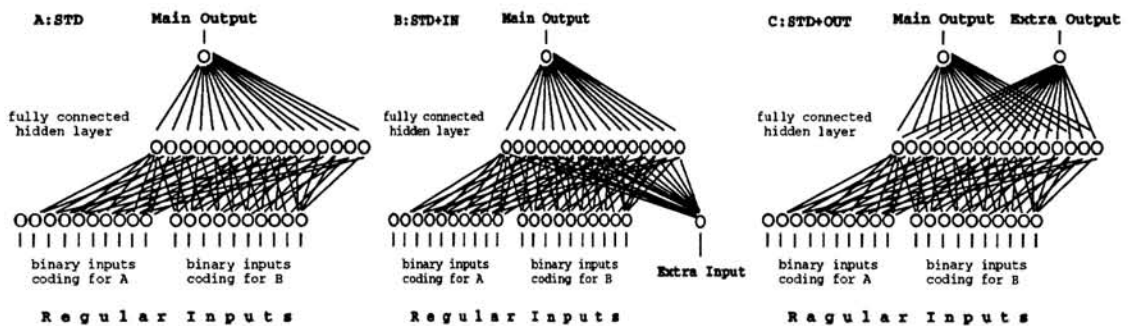

Figure 1: Three Neural Net Architectures for Learning F1

Backpropagation is done with per-epoch updating and early stopping. Each trial uses new random training, halt, and test sets. Training sets contain 50 patterns. This is enough data to get good performance, but not so much that there is not room for improvement. We use large halt and test sets — 1000 cases each — to minimize the effect of sampling error in the measured performances. Larger halt and test sets yield similar results. We use this methodology for all the experiments in this paper.

Table 1 shows the mean performance of 50 trials of STD Net 1a with backpropagation and early stopping.

Now consider a similar function:

F2(A,B) = SIGMOID(A-B).

Suppose, in addition to the 10-bit codings for A and B, you are given the unencoded unary value F2(A,B) as an extra input feature. Will this extra input help you learn F1(A,B) better? Probably not. A+B and A-B do not correlate for random A and B. The correlation coefficient for our training sets is typically less than ±0.01. Because

Table 1: Mean Test Set Root-Mean-Squared-Error on F1

| Network | Trials | Mean RMSE | Significance |
|---------|--------|-----------|--------------|
| STD | 50 | 0.0648 | - |
| STD+IN | 50 | 0.0647 | ns |
| STD+OUT | 50 | 0.0631 | 0.013* |

of this, knowing the value of F2(A,B) does not tell you much about the target value F1(A,B) (and vice-versa).

F1(A,B)'s poor correlation with F2(A,B) hurts backprop's ability to learn to use F2(A,B) to predict F1(A,B). The STD+IN net in Figure 1b has 21 inputs — 20 for the binary codes for A and B, and an extra input for F2(A,B). The 2nd line in Table 1 shows the performance of STD+IN for the same training, halting, and test sets used by STD; the only difference is that there is an extra input feature in the data sets for STD+IN. Note that the performance of STD+IN is not significantly different from that of STD — the extra information contained in the feature F2(A,B) does not help backpropagation learn F1(A,B) *when used as an extra input.*

If F2(A,B) does not help backpropagation learn F1(A,B) when used as an input, should we ignore it altogether? No. F1(A,B) and F2(A,B) are strongly related. They both benefit from decoding the binary input encoding to compute the subfeatures A and B. If, instead of using F2(A,B) as an extra input, it is used as an extra output trained with backpropagation, it will bias the shared hidden layer to learn A and B better, and this will help the net better learn to predict F1(A,B).

Figure 1c shows a net with 20 inputs for A and B, and 2 outputs, one for F1(A,B) and one for F2(A,B). Error is back-propagated from both outputs, but the performance of this net is evaluated only on the output F1(A,B) and early stopping is done using only the performance of this output. The 3rd line in Table 1 shows the mean performance of 50 trials of this multitask net on F1(A,B). Using F2(A,B) as an extra output significantly improves performance on F1(A,B). Using the extra feature as an extra output is better than using it as an extra input. *By using F2(A,B) as an output we make use of more than just the individual output values F2(A,B) but learn to extract information about the function mapping the inputs to F2(A,B). This is a key difference between using features as inputs and outputs.*

The increased performance of STD+OUT over STD and STD+IN is not due to STD+OUT reducing the capacity available for the main task F1(). All three nets — STD, STD+IN, STD+OUT — perform better with *more* hidden units. (Because larger capacity favors STD+OUT over STD and STD+IN, we report results for the moderate sized 16 hidden unit nets to be fair to STD and STD+IN.)

## 3   Noisy Features

This section presents two problems where extra features are more useful as inputs if they have low noise, but which become more useful as outputs as their noise increases. Because the extra features are ideal features for these problems, this demonstrates that what we observed in the previous section does not depend on the extra features being contrived so that their correlation with the main task is low – features with high correlation can still be more useful as outputs.

Once again, consider the main task from the previous section:

F1(A,B) = SIGMOID(A+B)

Now consider these extra features:

EF(A) = A + NOISE_SCALE * Noise1

EF(B) = B + NOISE_SCALE * Noise2

Noise1 and Noise2 are uniformly sampled on [-1,1]. If NOISE_SCALE is not too large, EF(A) and EF(B) are excellent input features for learning F1(A,B) because the net can avoid learning to decode the binary input representations. However, as NOISE_SCALE increases, EF(A) and EF(B) become less useful and it is better for the net to learn F1(A,B) from the binary inputs for A and B.

As before, we try using the extra features as either extra inputs or as extra outputs. Again, the training sets have 50 patterns, and the halt and test sets have 1000 patterns. Unlike before, however, we ran preliminary tests to find the best net size. The results showed 256 hidden units to be about optimal for the STD nets with early stopping on this problem.

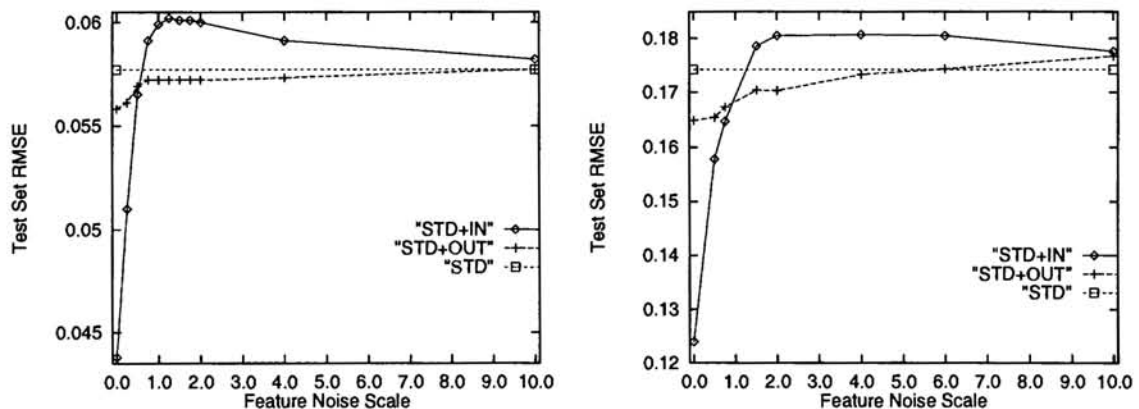

Figure 2: STD, STD+IN, and STD+OUT on F1 (left) and F3 (right)

Figure 2a plots the average performance of 50 trials of STD+IN and STD+OUT as NOISE_SCALE varies from 0.0 to 10.0. The performance of STD, which does not use EF(A) and EF(B), is shown as a horizontal line; it is independent of NOISE_SCALE. Let's first examine the results of STD+IN which uses EF(A) and EF(B) as extra inputs. As expected, when the noise is small, using EF(A) and EF(B) as extra inputs improves performance considerably. As the noise increases, however, this improvement decreases. Eventually there is so much noise in EF(A) and EF(B) that they no longer help the net if used as inputs. And, if the noise increases further, using EF(A) and EF(B) as extra inputs actually hurts. Finally, as the noise gets very large, performance asymptotes back towards the baseline.

Using EF(A) and EF(B) as extra outputs yields quite different results. When the noise is low, they do not help as much as they did as extra inputs. As the noise increases, however, at some point they help more as extra outputs than as extra inputs, and never hurt performance the way the noisy extra inputs did.

Why does noise cause STD+IN to perform worse than STD? With a finite training sample, correlations between noisy inputs and the main task cause the network to use the noisy inputs. To the extent that the main task is a function of the noisy inputs, it must pass the noise to the output, causing the output to be noisy. Also, as the net comes to depend on the noisy inputs, it depends less on the noise-free binary inputs. The noisy inputs *explain away* some of the training signal, so less is available to encourage learning to decode the binary inputs.

Why does noise not hurt STD+OUT as much as it hurts STD+IN? As outputs, the net is learning the mapping from the regular inputs to EF(A) and EF(B). Early in training, the net learns to interpolate through the noise and thus learns smooth functions for EF(A) and EF(B) that have reasonable fidelity to the true mapping. This makes learning less sensitive to the noise added to these features.

## 3.1 Another Problem

F1(A,B) is only mildly nonlinear because A and B do not go far into the tails of the SIGMOID. Do the results depend on this smoothness? To check, we modified F1(A,B) to make it more nonlinear. Consider this function:

F3(A,B) = SIGMOID(EXPAND(SIGMOID(A)–SIGMOID(B)))

where EXPAND scales the inputs from (SIGMOID(A)–SIGMOID(B)) to the range [-12.5,12.5], and A and B are drawn from [-12.5,12.5]. F3(A,B) is significantly more nonlinear than F1(A,B) because the expanded scales of A and B, and expanding the difference to [-12.5,12.5] before passing it through another sigmoid, cause much of the data to fall in the tails of either the inner or outer sigmoids.

Consider these extra features:

EF(A) = SIGMOID(A) + NOISE_SCALE * Noise1

EF(B) = SIGMOID(B) + NOISE_SCALE * Noise2

where Noises are sampled as before. Figure 2B shows the results of using extra features EF(A) and EF(B) as extra inputs or as extra outputs. The trend is similar to that in Figure 2A but the benefit of STD+OUT is even larger at low noise. The data for 2a and 2b are generated using different seeds, 2a used steepest descent and Mitre's Aspirin simulator, 2b used conjugate gradient and Toronto's Xerion simulator, and F1 and F3 do not behave as similarly as their definitions might suggest. The similarity between the two graphs is due to the ubiquity of the phenomena, not to some small detail of the test functions or how the experiments were run.

## 4 A Classification Problem

This section presents a problem that combines feature correlation (Section 1) and feature noise (Section 2) into one problem. Consider the 1-D classification problem, shown in Figure 3, of separating two Gaussian distributions with means 0 and 1, and standard deviations of 1. This problem is simple to learn if the 1-D input is coded as a single, continuous input but can be made harder by embedding it non-linearly in a higher dimensional space. Consider encoding input values defined on [0.0,15.0] using an *interpolated* 4-D Gray code($\overline{GC}$); integer values are mapped to a 4-D binary Gray code and intervening non-integers are mapped linearly to intervening 4-D vectors between the binary Gray codes for the bounding integers. As the Gray code flips only one bit between neighboring integers this involves simply interpolating along the 1 dimension in the 4-D unit cube that changes. Thus 3.4 is encoded as $.4(\overline{GC}(4) - \overline{GC}(3)) + \overline{GC}(3)$.

The extra feature is a 1-D value correlated (with correlation $\rho$) with the original unencoded regular input, X. The extra feature is drawn from a Gaussian distribution with mean $\rho \times (X - .5) + .5$ and standard deviation $\sqrt{(1 - \rho^2)}$. Examples of the distributions of the unencoded original dimension and the extra feature for various correlations are shown in Figure 3. This problem has been carefully constructed so that the optimal classification boundary does not change as $\rho$ varies.

Consider the extreme cases. At $\rho = 1$, the extra feature is exactly an unencoded

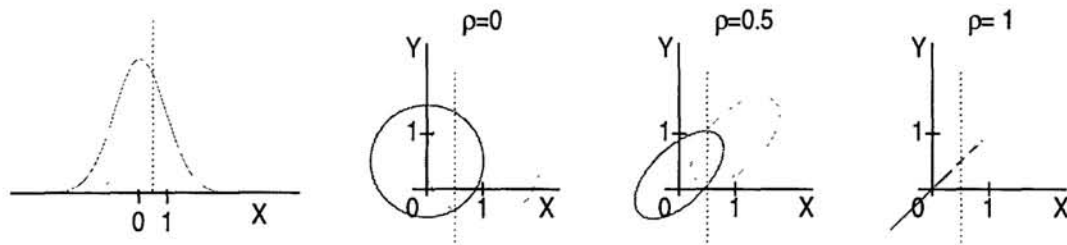

Figure 3: Two Overlapped Gaussian Classes (left), and An Extra Feature (y-axis) Correlated Different Amounts ($\rho = 0$: no correlation, $\rho = 1$: perfect correlation) With the unencoded version of the Regular Input (x-axis)

version of the regular input. A STD+IN net using this feature as an extra input could ignore the encoded inputs and solve the problem using this feature alone. An STD+OUT net using this extra feature as an extra output would have its hidden layer biased towards representations that decode the Gray code, which is useful to the main classification task. At the other extreme ($\rho = 0$), we expect nets using the extra feature to learn no better than one using just the regular inputs because there is no useful information provided by the uncorrelated extra feature. The interesting case is between the two extremes. We can imagine a situation where as an output, the extra feature is still able to help STD+OUT by guiding it to decode the Gray code but does not help STD+IN because of the high level of noise.

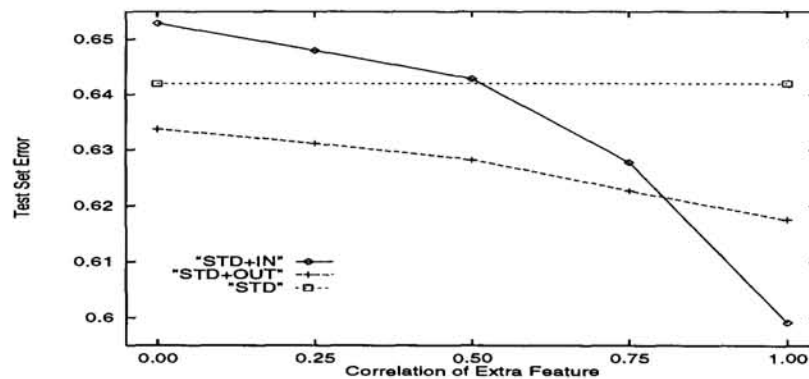

Figure 4: STD, STD+IN, and STD+OUT vs. $\rho$ on the Classification Problem

The class output unit uses a sigmoid transfer function and cross-entropy error measure. The output unit for the correlated extra feature uses a linear transfer function and squared error measure. Figure 4 shows the average performance of 50 trials of STD, STD+IN, and STD+OUT as a function of $\rho$ using networks with 20 hidden units, 70 training patterns, and halt and test sets of 1000 patterns each. As in the previous section, STD+IN is much more sensitive to changes in the extra feature than STD+OUT, so that by $\rho = 0.75$ the curves cross and for $\rho$ less than 0.75, the dimension is actually more useful as an output dimension than an extra input.

## 5   Discussion

Are the benefits of using some features as extra outputs instead of as inputs large enough to be interesting? Yes. Using only 1 or 2 features as extra outputs instead of as inputs reduced error 2.5% on the problem in Section 1, more than 5% in regions of the graphs in Section 2, and more than 2.5% in regions of the graph in Section

3. In domains where many features might be moved, the net effect may be larger.

Are some features more useful as outputs than as inputs only on contrived problems? No. In this paper we used the simplest problems we could devise where a few features worked better as outputs than as inputs. But our findings explain a result we noted previously, but did not understand, when applying multitask learning to pneumonia risk prediction[4]. There, we had the choice of using lab tests that would be unavailable on future patients as extra outputs, or using poor — i.e., noisy — predictions of them as extra inputs. Using the lab tests as extra outputs worked better. If one compares the zero noise points for STD+OUT (there's no noise in a feature when used as an output because we use the values in the training set, not predicted values) with the high noise points for STD+IN in the graphs in Section 2, it is easy to see why STD+OUT could perform much better.

This paper shows that the benefit of using a feature as an extra output is different from the benefit of using that feature as an input. As an input, the net has access to the values on the training and test cases to use for prediction. As an output, however, the net is instead biased to learn a mapping from the other inputs in the training set to that output. From the graphs it is clear that some features help when used either as an input, or as an output. Given that the benefit of using a feature as an extra output is different from that of using it as an input, can we get both benefits? Our early results with techniques that reap both benefits by allowing some features to be used simultaneously as both inputs and outputs while preventing learning direct feedthrough identity mappings are promising.

## Acknowledgements

R. Caruana was supported in part by ARPA grant F33615-93-1-1330, NSF grant BES-9315428, and Agency for Health Care Policy and Research grant HS06468. V. de Sa was supported by postdoctoral fellowships from NSERC (Canada) and the Sloan Foundation. We thank Mitre Group for the Aspirin/Migraines Simulator and The University of Toronto for the Xerion Simulator.

## References

[1] Y.S. Abu-Mostafa, "Learning From Hints in Neural Networks," *Journal of Complexity* 6:2, pp. 192–198, 1989.

[2] S. Baluja, and D.A. Pomerleau, "Using the Representation in a Neural Network's Hidden Layer for Task-Specific Focus of Attention". In C. Mellish (ed.) The International Joint Conference on Artificial Intelligence 1995 (IJCAI-95): Montreal, Canada. IJCAII & Morgan Kaufmann. San Mateo, CA. pp 133-139, 1995.

[3] S. Becker and G. E. Hinton, "A self-organizing neural network that discovers surfaces in random-dot stereograms," *Nature* **355** pp. 161-163, 1992.

[4] R. Caruana, S. Baluja, and T. Mitchell, "Using the Future to Sort Out the Present: Rankprop and Multitask Learning for Pneumnia Risk Prediction," *Advances in Neural Information Processing Systems 8*, 1996.

[5] R. Caruana, "Learning Many Related Tasks at the Same Time With Backpropagation," *Advances in Neural Information Processing Systems 7*, 1995.

[6] V. R. de Sa, "Learning classification with unlabeled data," *Advances in Neural Information Processing Systems 6*, pp. 112–119, Morgan Kaufmann, 1994.

[7] P. Simard, B. Victorri, Y. L. Cun, and J. Denker, "Tangent prop — a formalism for specifying selected invariances in an adaptive network," *Advances in Neural Information Processing Systems 4*, pp. 895–903, Morgan Kaufmann, 1992.

[8] S. Thrun and T. Mitchell, "Learning One More Thing," CMU TR: CS-94-184, 1994.